# Planning for Markov Decision Processes with Sparse Stochasticity

**Maxim Likhachev**
School of Computer Science
Carnegie Mellon University
Pittsburgh, PA 15213
maxim+@cs.cmu.edu

**Geoff Gordon**
School of Computer Science
Carnegie Mellon University
Pittsburgh, PA 15213
ggordon@cs.cmu.edu

**Sebastian Thrun**
Dept. of Computer Science
Stanford University
Stanford CA 94305
thrun@stanford.edu

## Abstract

Planning algorithms designed for deterministic worlds, such as A* search, usually run much faster than algorithms designed for worlds with uncertain action outcomes, such as value iteration. Real-world planning problems often exhibit uncertainty, which forces us to use the slower algorithms to solve them. Many real-world planning problems exhibit *sparse* uncertainty: there are long sequences of deterministic actions which accomplish tasks like moving sensor platforms into place, interspersed with a small number of sensing actions which have uncertain outcomes. In this paper we describe a new planning algorithm, called MCP (short for MDP Compression Planning), which combines A* search with value iteration for solving Stochastic Shortest Path problem in MDPs with sparse stochasticity. We present experiments which show that MCP can run substantially faster than competing planners in domains with sparse uncertainty; these experiments are based on a simulation of a ground robot cooperating with a helicopter to fill in a partial map and move to a goal location.

In deterministic planning problems, optimal paths are acyclic: no state is visited more than once. Because of this property, algorithms like A* search can guarantee that they visit each state in the state space no more than once. By visiting the states in an appropriate order, it is possible to ensure that we know the exact value of all of a state's possible successors before we visit that state; so, the first time we visit a state we can compute its correct value.

By contrast, if actions have uncertain outcomes, optimal paths may contain cycles: some states will be visited two or more times with positive probability. Because of these cycles, there is no way to order states so that we determine the values of a state's successors before we visit the state itself. Instead, the only way to compute state values is to solve a set of simultaneous equations.

In problems with sparse stochasticity, only a small fraction of all states have uncertain outcomes. It is these few states that cause all of the cycles: while a deterministic state $s$ may participate in a cycle, the only way it can do so is if one of its successors has an action with a stochastic outcome (and only if this stochastic action can lead to a predecessor of $s$).

In such problems, we would like to build a smaller MDP which contains only states which are related to stochastic actions. We will call such an MDP a *compressed MDP*, and we will call its states *distinguished states*. We could then run fast algorithms like A* search to plan paths between distinguished states, and reserve slower algorithms like value iteration for deciding how to deal with stochastic outcomes.

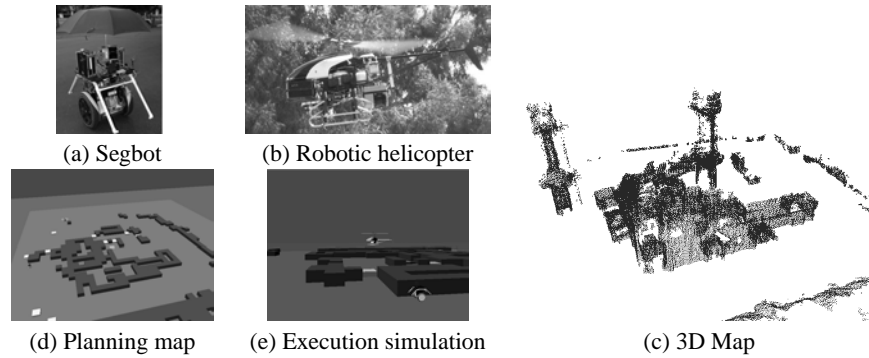

(a) Segbot           (b) Robotic helicopter

(d) Planning map    (e) Execution simulation        (c) 3D Map

Figure 1: Robot-Helicopter Coordination

There are two problems with such a strategy. First, there can be a large number of states which are related to stochastic actions, and so it may be impractical to enumerate all of them and make them all distinguished states; we would prefer instead to distinguish only states which are likely to be encountered while executing some policy which we are considering. Second, there can be a large number of ways to get from one distinguished state to another: edges in the compressed MDP correspond to sequences of actions in the original MDP. If we knew the values of all of the distinguished states exactly, then we could use A* search to generate optimal paths between them, but since we do not we cannot.

In this paper, we will describe an algorithm which incrementally builds a compressed MDP using a sequence of deterministic searches. It adds states and edges to the compressed MDP only by encountering them along trajectories; so, it never adds irrelevant states or edges to the compressed MDP. Trajectories are generated by deterministic search, and so undistinguished states are treated only with fast algorithms. Bellman errors in the values for distinguished states show us where to try additional trajectories, and help us build the relevant parts of the compressed MDP as quickly as possible.

## 1 Robot-Helicopter Coordination Problem

The motivation for our research was the problem of coordinating a ground robot and a helicopter. The ground robot needs to plan a path from its current location to a goal, but has only partial knowledge of the surrounding terrain. The helicopter can aid the ground robot by flying to and sensing places in the map.

Figure 1(a) shows our ground robot, a converted Segway with a SICK laser rangefinder. Figure 1(b) shows the helicopter, also with a SICK. Figure 1(c) shows a 3D map of the environment in which the robot operates. The 3D map is post-processed to produce a discretized 2D environment (Figure 1(d)). Several places in the map are unknown, either because the robot has not visited them or because their status may have changed (e.g, a car may occupy a driveway). Such places are shown in Figure 1(d) as white squares. The elevation of each white square is proportional to the probability that there is an obstacle there; we assume independence between unknown squares.

The robot must take the unknown locations into account when planning for its route. It may plan a path through these locations, but it risks having to turn back if its way is blocked. Alternately, the robot can ask the helicopter to fly to any of these places and sense them. We assign a cost to running the robot, and a somewhat higher cost to running the helicopter. The planning task is to minimize the expected overall cost of running the robot and the helicopter while getting the robot to its destination and the helicopter back to its home base. Figure 1(e) shows a snapshot of the robot and helicopter executing a policy.

Designing a good policy for the robot and helicopter is a POMDP planning problem; unfortunately POMDPs are in general difficult to solve (PSPACE-complete [7]). In the POMDP representation, a state is the position of the robot, the current location of the helicopter (a point on a line segment from one of the unknown places to another unknown place or the home base), and the true status of each unknown location. The positions of the robot and the helicopter are observable, so that the only hidden variables are whether each

unknown place is occupied. The number of states is (# of robot locations)×(# of helicopter locations)×$2^{\text{\# of unknown places}}$. So, the number of states is exponential in the number of unknown places and therefore quickly becomes very large.

We approach the problem by planning in the belief state space, that is, the space of probability distributions over world states. This problem is a continuous-state MDP; in this belief MDP, our state consists of the ground robot's location, the helicopter's location, and a probability of occupancy for each unknown location. We will discretize the continuous probability variables by breaking the interval $[0, 1]$ into several chunks; so, the number of belief states is exponential in the number of unknown places, and classical algorithms such as value iteration are infeasible even on small problems.

If sensors are perfect, this domain is acyclic: after we sense a square we know its true state forever after. On the other hand, imperfect sensors can lead to cycles: new sensor data can contradict older sensor data and lead to increased uncertainty. With or without sensor noise, our belief state MDP differs from general MDPs because its stochastic transitions are sparse: large portions of the policy (while the robot and helicopter are traveling between unknown locations) are deterministic. The algorithm we propose in this paper takes advantage of this property of the problem, as we explain in the next section.

## 2  The Algorithm

Our algorithm can be broken into two levels. At a high level, it constructs a *compressed MDP*, denoted $M^c$, which contains only the start, the goal, and some states which are outcomes of stochastic actions. At a lower level, it repeatedly runs deterministic searches to find new information to put into $M^c$. This information includes newly-discovered stochastic actions and their outcomes; better deterministic paths from one place to another; and more accurate value estimates similar to Bellman backups. The deterministic searches can use an admissible heuristic $h$ to focus their effort, so we can often avoid putting many irrelevant actions into $M^c$.

Because $M^c$ will often be much smaller than $M$, we can afford to run stochastic planning algorithms like value iteration on it. On the other hand, the information we get by planning in $M^c$ will improve the heuristic values that we use in our deterministic searches; so, the deterministic searches will tend to visit only relevant portions of the state space.

### 2.1  Constructing and Solving a Compressed MDP

Each action in the compressed MDP represents several consecutive actions in $M$: if we see a sequence of states and actions $s_1, a_1, s_2, a_2, \ldots, s_k, a_k$ where $a_1$ through $a_{k-1}$ are deterministic but $a_k$ is stochastic, then we can represent it in $M^c$ with a single action $a$, available at $s_1$, whose outcome distribution is $P(s' \mid s_k, a_k)$ and whose cost is

$$c(s_1, a, s') = \sum_{i=1}^{k-1} c(s_i, a_i, s_{i+1}) + c(s_k, a_k, s')$$

(See Figure 2(a) for an example of such a compressed action.) In addition, if we see a sequence of deterministic actions ending in $s_{\text{goal}}$, say $s_1, a_1, s_2, a_2, \ldots, s_k, a_k, s_{k+1} = s_{\text{goal}}$, we can define a compressed action which goes from $s_1$ to $s_{\text{goal}}$ at cost $\sum_{i=1}^{k} c(s_i, a_i, s_{i+1})$. We can label each compressed action that starts at $s$ with $(s, s', a)$ (where $a = \textbf{null}$ if $s' = s_{\text{goal}}$).

Among all compressed actions starting at $s$ and ending at $(s', a)$ there is (at least) one with lowest expected cost; we will call such an action an *optimal compression* of $(s, s', a)$. Write $A_{\text{stoch}}$ for the set of all pairs $(s, a)$ such that action $a$ when taken from state $s$ has more than one possible outcome, and include as well $(s_{\text{goal}}, \textbf{null})$. Write $S_{\text{stoch}}$ for the states which are possible outcomes of the actions in $A_{\text{stoch}}$, and include $s_{\text{start}}$ as well. If we include in our compressed MDP an optimal compression of $(s, s', a)$ for every $s \in S_{\text{stoch}}$ and every $(s', a) \in A_{\text{stoch}}$, the result is what we call the *full compressed MDP*; an example is in Figure 2(b).

If we solve the full compressed MDP, the value of each state will be the same as the value of the corresponding state in $M$. However, we do not need to do that much work:

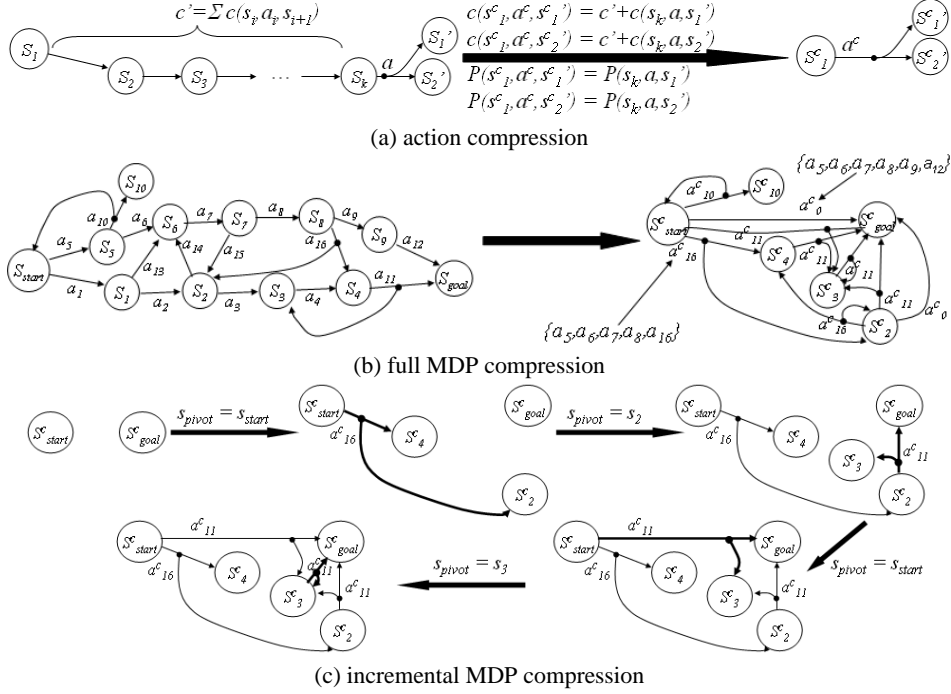

(a) action compression

(b) full MDP compression

(c) incremental MDP compression

Figure 2: MDP compression

**Main()**
01 initialize $M^c$ with $s_{\text{start}}$ and $s_{\text{goal}}$ and set their $v$-values to 0;
02 while ($\exists s \in M^c$ s.t. $RHS(s) - v(s) > \delta$ and $s$ belongs to the current greedy policy)
03     select $s_{\text{pivot}}$ to be any such state $s$;
04     $[v; v_{\text{lim}}]$ = Search($s_{\text{pivot}}$);
05     $v(s_{\text{pivot}}) = v$;
06     set the cost $c(s_{\text{pivot}}, \bar{a}, s_{\text{goal}})$ of the limit action $\bar{a}$ from $s_{\text{pivot}}$ to $v_{\text{lim}}$;
07     optionally run some algorithm satisfying req. A for a bounded amount of time to improve the value function in $M^c$;

Figure 3: MCP main loop

many states and actions in the full compressed MDP are irrelevant since they do not appear in the optimal policy from $s_{\text{start}}$ to $s_{\text{goal}}$. So, the goal of the MCP algorithm will be to construct only the relevant part of the compressed MDP by building $M^c$ incrementally. Figure 2(c) shows the incremental construction of a compressed MDP which contains all of the stochastic states and actions along an optimal policy in $M$.

The pseudocode for MCP is given in Figure 3. It begins by initializing $M^c$ to contain only $s_{\text{start}}$ and $s_{\text{goal}}$, and it sets $v(s_{\text{start}}) = v(s_{\text{goal}}) = 0$. It maintains the invariant that $0 \le v(s) \le v^*(s)$ for all $s$. On each iteration, MCP looks at the Bellman error of each of the states in $M^c$. The Bellman error is $v(s) - RHS(s)$, where

$$RHS(s) = \min_{a \in A(s)} RHS(s, a) \qquad RHS(s, a) = E_{s' \in \text{succ}(s,a)}(c(s, a, s') + v(s'))$$

By convention the min of an empty set is $\infty$, so an $s$ which does not have any compressed actions yet is considered to have infinite $RHS$.

MCP selects a state with negative Bellman error, $s_{\text{pivot}}$, and starts a search at that state. (We note that there exist many possible ways to select $s_{\text{pivot}}$; for example, we can choose the state with the largest negative Bellman error, or the error when weighted by state visitation probabilities in the best policy in $M^c$.) The goal of this search is to find a new compressed action $a$ such that its $RHS$-value can provide a new lower bound on $v^*(s_{\text{pivot}})$. This action can either decrease the current $RHS(s_{\text{pivot}})$ (if $a$ seems to be a better action in terms of the current $v$-values of action outcomes) or prove that the current $RHS(s_{\text{pivot}})$ is valid. Since $v(s') \le v^*(s')$, one way to guarantee that $RHS(s_{\text{pivot}}, a) \le v^*(s_{\text{pivot}})$ is

to compute an optimal compression of $(s_{\text{pivot}}, s, a)$ for all $s, a$, then choose the one with the smallest *RHS*. A more sophisticated strategy is to use an A* search with appropriate safeguards to make sure we never overestimate the value of a stochastic action. MCP, however, uses a modified A* search which we will describe in the next section.

As the search finds new compressed actions, it adds them and their outcomes to $M^c$. It is allowed to initialize newly-added states to any admissible values. When the search returns, MCP sets $v(s_{\text{pivot}})$ to the returned value. This value is at least as large as $RHS(s_{\text{pivot}})$. Consequently, Bellman error for $s_{\text{pivot}}$ becomes non-negative.

In addition to the compressed action and the updated value, the search algorithm returns a "limit value" $v_{\text{lim}}(s_{\text{pivot}})$. These limit values allow MCP to run a standard MDP planning algorithm on $M^c$ to improve its $v(s)$ estimates. MCP can use any planning algorithm which guarantees that, for any $s$, it will not lower $v(s)$ and will not increase $v(s)$ beyond the smaller of $v_{\text{lim}}(s)$ and $RHS(s)$ (**Requirement A**). For example, we could insert a fake "limit action" into $M^c$ which goes directly from $s_{\text{pivot}}$ to $s_{\text{goal}}$ at cost $v_{\text{lim}}(s_{\text{pivot}})$ (as we do on line 06 in Figure 3), then run value iteration for a fixed amount of time, selecting for each backup a state with negative Bellman error.

After updating $M^c$ from the result of the search and any optional planning, MCP begins again by looking for another state with a negative Bellman error. It repeats this process until there are no negative Bellman errors larger than $\delta$. For small enough $\delta$, this property guarantees that we will be able to find a good policy (see section 2.3).

## 2.2   Searching the MDP Efficiently

The top level algorithm (Figure 3) repeatedly invokes a search method for finding trajectories from $s_{\text{pivot}}$ to $s_{\text{goal}}$. In order for the overall algorithm to work correctly, there are several properties that the search must satisfy. First, the estimate $v$ that search returns for the expected cost of $s_{\text{pivot}}$ should always be admissible. That is, $0 \leq v \leq v^*(s_{\text{pivot}})$ (**Property 1**). Second, the estimate $v$ should be no less than the one-step lookahead value of $s_{\text{pivot}}$ in $M^c$. That is, $v \geq RHS(s_{\text{pivot}})$ (**Property 2**). This property ensures that search either increases the value of $s_{\text{pivot}}$ or finds additional (or improved) compressed actions. The third and final property is for the $v_{\text{lim}}$ value, and it is only important if MCP uses its optional planning step (line 07). The property is that $v \leq v_{\text{lim}} \leq \overline{v^*}(s_{\text{pivot}})$ (**Property 3**). Here $\overline{v^*}(s_{\text{pivot}})$ denotes the minimum expected cost of starting at $s_{\text{pivot}}$, picking a compressed action *not* in $M^c$, and acting optimally from then on. (Note that $\overline{v^*}$ can be larger than $v^*$ if the optimal compressed action is already part of $M^c$.) Property 3 uses $\overline{v^*}$ rather than $v^*$ since the latter is not known while it is possible to compute a lower bound on the former efficiently (see below).

One could adapt A* search to satisfy at least Properties 1 and 2 by assuming that we can control the outcome of stochastic actions. However, this sort of search is highly optimistic and can bias the search towards improbable trajectories. Also, it can only use heuristics which are even more optimistic than it is: that is, $h$ must be admissible with respect to the optimistic assumption of controlled outcomes.

We therefore present a version of A*, called MCP-search (Figure 4), that is more efficient for our purposes. MCP-search finds the correct expected value for the first stochastic action it encounters on any given trajectory, and is therefore far less optimistic. And, MCP-search only requires heuristic values to be admissible with respect to $v^*$ values, $h(s) \leq v^*(s)$. Finally, MCP-search speeds up repetitive searches by improving heuristic values based on previous searches.

A* maintains a priority queue, *OPEN*, of states which it plans to expand. The *OPEN* queue is sorted by $f(s) = g(s) + h(s)$, so that A* always expands next a state which appears to be on the shortest path from start to goal. During each expansion a state $s$ is removed from *OPEN* and all the $g$-values of $s$'s successors are updated; if $g(s')$ is decreased for some state $s'$, A* inserts $s'$ into *OPEN*. A* terminates as soon as the goal state is expanded. We use the variant of A* with pathmax [5] to use efficiently heuristics that do not satisfy the triangle inequality.

MCP is similar to A*, but the *OPEN* list can also contain state-action pairs $\{s, a\}$ where $a$ is a stochastic action (line 31). Plain states are represented in *OPEN* as $\{s, \textbf{null}\}$. Just

**ImproveHeuristic($s$)**
01 if $s \in M^c$ then $h(s) = \max(h(s), v(s))$;
02 improve heuristic $h(s)$ further if possible using $fbest$ and $g(s)$ from previous iterations;

**procedure fvalue($\{s, a\}$)**
03 if $s = $ **null** return $\infty$;
04 else if $a = $ **null** return $g(s) + h(s)$;
05 else return $g(s) + \max(h(s), E_{s' \in Succ(s,a)}\{c(s, a, s') + h(s')\})$;

**CheckInitialize($s$)**
06 if $s$ was accessed last in some previous search iteration
07   ImproveHeuristic($s$);
08 if $s$ was not yet initialized in the current search iteration
09   $g(s) = \infty$;

**InsertUpdateCompAction($s_{\text{pivot}}$, $s$, $a$)**
10 reconstruct the path from $s_{\text{pivot}}$ to $s$;
11 insert compressed action $(s_{\text{pivot}}, s, a)$ into $A(s_{\text{pivot}})$ (or update the cost if a cheaper path was found)
12 for each outcome $u$ of $a$ that was not in $M^c$ previously
13   set $v(u)$ to $h(u)$ or any other value less than or equal to $v^*(u)$;
14   set the cost $c(u, \bar{a}, s_{\text{goal}})$ of the limit action $\bar{a}$ from $u$ to $v(u)$;

**procedure Search($s_{\text{pivot}}$)**
15 CheckInitialize($s_{\text{goal}}$), CheckInitialize($s_{\text{pivot}}$);
16 $g(s_{\text{pivot}}) = 0$;
17 $OPEN = \{\{s_{\text{pivot}}, \textbf{null}\}\}$;
18 $\{s_{\text{best}}, a_{\text{best}}\} = \{\textbf{null}, \textbf{null}\}$, $fbest = \infty$;
19 while($g(s_{\text{goal}}) > \min_{\{s,a\} \in OPEN}(\text{fvalue}(\{s, a\}))$ AND $fbest + \theta > \min_{\{s,a\} \in OPEN}(\text{fvalue}(\{s, a\}))$)
20   remove $\{s, a\}$ with the smallest fvalue($\{s, a\}$) from $OPEN$ breaking ties towards the pairs with $a = $ **null**;
21   if $a = $ **null**   //expand state $s$
22    for each $s' \in Succ(s)$
23     CheckInitialize($s'$);
24    for each deterministic $a' \in A(s)$
25     $s' = Succ(s, a')$;
26     $h(s') = \max(h(s'), h(s) - c(s, a', s'))$;
27     if $g(s') > g(s) + c(s, a', s')$
28      $g(s') = g(s) + c(s, a', s')$;
29      insert/update $\{s', \textbf{null}\}$ into $OPEN$ with fvalue($\{s', \textbf{null}\}$);
30    for each stochastic $a' \in A(s)$
31     insert/update $\{s, a'\}$ into $OPEN$ with fvalue($\{s, a'\}$);
32   else     //encode stochastic action $a$ from state $s$ as a compressed action from $s_{\text{pivot}}$
33    InsertUpdateCompAction($s_{\text{pivot}}$, $s$, $a$);
34    if $fbest > \text{fvalue}(\{s, a\})$ then $\{s_{\text{best}}, a_{\text{best}}\} = \{s, a\}$, $fbest = \text{fvalue}(\{s, a\})$;
35 if $(g(s_{\text{goal}}) \leq \min_{\{s,a\} \in OPEN}(\text{fvalue}(\{s, a\}))$ AND $OPEN \neq \emptyset)$
36   reconstruct the path from $s_{\text{pivot}}$ to $s_{\text{goal}}$;
37   update/insert into $A(s_{\text{pivot}})$ a deterministic action $a$ leading to $s_{\text{goal}}$;
38   if $fbest \geq g(s_{\text{goal}})$ then $\{s_{\text{best}}, a_{\text{best}}\} = \{s_{\text{goal}}, \textbf{null}\}$, $fbest = g(s_{\text{goal}})$;
39 return $[fbest; \min_{\{s,a\} \in OPEN}(\text{fvalue}(\{s, a\}))]$;

Figure 4: MCP-search Algorithm

like A\*, MCP-search expands elements in the order of increasing $f$-values, but it breaks ties towards elements encoding plain states (line 20). The $f$-value of $\{s, a\}$ is defined as $g(s) + \max[h(s), E_{s' \in Succ(s,a)}(c(s, a, s') + h(s'))]$ (line 05). This $f$-value is a lower bound on the cost of a policy that goes from $s_{\text{start}}$ to $s_{\text{goal}}$ by first executing a series of deterministic actions until action $a$ is executed from state $s$. This bound is as tight as possible given our heuristic values.

State expansion (lines 21-31) is very similar to A\*. When the search removes from $OPEN$ a state-action pair $\{s, a\}$ with $a \neq \textbf{null}$, it adds a compressed action to $M^c$ (line 33). It also adds a compressed action if there is an optimal deterministic path to $s_{\text{goal}}$ (line 37). $fbest$ tracks the minimum $f$-value of all the compressed actions found. As a result, $fbest \leq v^*(s_{\text{pivot}})$ and is used as a new estimate for $v(s_{\text{pivot}})$. The limit value $v_{\text{lim}}(s_{\text{pivot}})$ is obtained by continuing the search until the minimum $f$-value of elements in $OPEN$ approaches $fbest + \theta$ for some $\theta \geq 0$ (line 19). This minimum $f$-value then provides a lower bound on $\overline{v^*}(s_{\text{pivot}})$.

To speed up repetitive searches, MCP-search improves the heuristic of every state that it encounters for the first time in the current search iteration (lines 01 and 02). Line 01 uses the fact that $v(s)$ from $M^c$ is a lower bound on $v^*(s)$. Line 02 uses the fact that $fbest - g(s)$ is a lower bound on $v^*(s)$ at the end of each previous call to **Search**; for more details see [4].

## 2.3 Theoretical Properties of the Algorithm

We now present several theorems about our algorithm. The proofs of these and other theorems can be found in [4]. The first theorem states the main properties of MCP-search.

**Theorem 1** *The search function terminates and the following holds for the values it returns:*

> *(a) if $s_{\text{best}} \neq \textbf{null}$ then $v^*(s_{\text{pivot}}) \geq fbest \geq E\{c(s_{\text{pivot}}, a_{\text{best}}, s') + v(s')\}$*
>
> *(b) if $s_{\text{best}} = \textbf{null}$ then $v^*(s_{\text{pivot}}) = fbest = \infty$*
>
> *(c) $fbest \leq \min_{\{s,a\} \in \text{OPEN}}(fvalue(\{s,a\})) \leq \overline{v^*}(s_{\text{pivot}})$.*

If neither $s_{\text{goal}}$ nor any state-action pairs were expanded, then $s_{\text{best}} = \textbf{null}$ and (b) says that there is no policy from $s_{\text{pivot}}$ that has a finite expected cost. Using the above theorem it is easy to show that MCP-search satisfies Properties 1, 2 and 3, considering that $fbest$ is returned as variable $v$ and $\min_{\{s,a\} \in OPEN}(\text{fvalue}(\{s,a\}))$ is returned as variable $v_{\text{lim}}$ in the main loop of the MCP algorithm (Figure 3). Property 1 follows directly from (a) and (b) and the fact that costs are strictly positive and $v$-values are non-negative. Property 2 also follows trivially from (a) and (b). Property 3 follows from (c). Given these properties the next theorem states the correctness of the outer MCP algorithm (in the theorem $\pi^c_{\text{greedy}}$ denotes a greedy policy that always chooses an action that looks best based on its cost and the $v$-values of its immediate successors).

**Theorem 2** *Given a deterministic search algorithm which satisfies Properties 1–3, the MCP algorithm will terminate. Upon termination, for every state $s \in M^c \cap \pi^c_{\text{greedy}}$ we have* $\text{RHS}(s) - \delta \leq v(s) \leq v^*(s)$.

Given the above theorem one can show that for $0 \leq \delta < c_{\text{min}}$ (where $c_{\text{min}}$ is the smallest expected action cost in our MDP) the expected cost of executing $\pi^c_{\text{greedy}}$ from $s_{\text{start}}$ is at most $\frac{c_{\text{min}}}{c_{\text{min}} - \delta} v^*(s_{\text{start}})$. Picking $\delta \geq c_{\text{min}}$ is not guaranteed to result in a proper policy, even though Theorem 2 continues to hold.

## 3 Experimental Study

We have evaluated the MCP algorithm on the robot-helicopter coordination problem described in section 1. To obtain an admissible heuristic, we first compute a value function for every possible configuration of obstacles. Then we weight the value functions by the probabilities of their obstacle configurations, sum them, and add the cost of moving the helicopter back to its base if it is not already there. This procedure results in optimistic cost estimates because it pretends that the robot will find out the obstacle locations immediately instead of having to wait to observe them.

The results of our experiments are shown in Figure 5. We have compared MCP against three algorithms: RTDP [1], LAO* [2] and value iteration on reachable states (VI). RTDP can cope with large size MDPs by focussing its planning efforts along simulated execution trajectories. LAO* uses heuristics to prune away irrelevant states, then repeatedly performs dynamic programming on the states in its current partial policy. We have implemented LAO* so that it reduces to AO* [6] when environments are acyclic (*e.g.*, the robot-helicopter problem with perfect sensing). VI was only able to run on the problems with perfect sensing since the number of reachable states was too large for the others.

The results support the claim that MCP can solve large problems with sparse stochasticity. For the problem with perfect sensing, on average MCP was able to plan 9.5 times faster than LAO*, 7.5 times faster than RTDP, and 8.5 times faster than VI. On average for these problems, MCP computed values for 58633 states while $M^c$ grew to 396 states, and MCP encountered 3740 stochastic transitions (to give a sense of the degree of stochasticity). The main cost of MCP was in its deterministic search subroutine; this fact suggests that we might benefit from anytime search techniques such as ARA* [3].

The results for the problems with imperfect sensing show that, as the number and density of uncertain outcomes increases, the advantage of MCP decreases. For these problems MCP was able to solve environments 10.2 times faster than LAO* but only 2.2 times faster than RTDP. On average MCP computed values for 127,442 states, while the size of $M^c$ was 3,713 states, and 24,052 stochastic transitions were encountered.

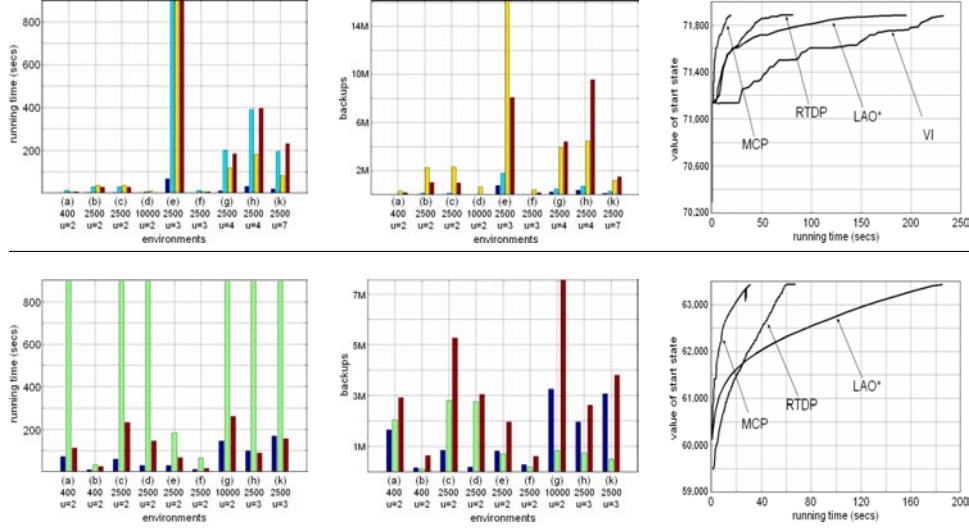

Figure 5: Experimental results. The top row: the robot-helicopter coordination problem with perfect sensors. The bottom row: the robot-helicopter coordination problem with sensor noise. Left column: running times (in secs) for each algorithm grouped by environments. Middle column: the number of backups for each algorithm grouped by environments. Right column: an estimate of the expected cost of an optimal policy ($v(s_{\text{start}})$) vs. running time (in secs) for experiment (k) in the top row and experiment (e) in the bottom row. Algorithms in the bar plots (left to right): MCP, LAO*, RTDP and VI (VI is only shown for problems with perfect sensing). The characteristics of the environments are given in the second and third rows under each of the bar plot. The second row indicates how many cells the 2D plane is discretized into, and the third row indicates the number of initially unknown cells in the environment.

## 4 Discussion

The MCP algorithm incrementally builds a compressed MDP using a sequence of deterministic searches. Our experimental results suggest that MCP is advantageous for problems with sparse stochasticity. In particular, MCP has allowed us to scale to larger environments than were otherwise possible for the robot-helicopter coordination problem.

### Acknowledgements

This research was supported by DARPA's MARS program. All conclusions are our own.

## References

[1] S. Bradtke A. Barto and S. Singh. Learning to act using real-time dynamic programming. *Artificial Intelligence*, 72:81–138, 1995.

[2] E. Hansen and S. Zilberstein. LAO*: A heuristic search algorithm that finds solutions with loops. *Artificial Intelligence*, 129:35–62, 2001.

[3] M. Likhachev, G. Gordon, and S. Thrun. ARA*: Anytime A* with provable bounds on sub-optimality. In *Advances in Neural Information Processing Systems (NIPS) 16*. Cambridge, MA: MIT Press, 2003.

[4] M. Likhachev, G. Gordon, and S. Thrun. MCP: Formal analysis. Technical report, Carnegie Mellon University, Pittsburgh, PA, 2004.

[5] L. Mero. A heuristic search algorithm with modifiable estimate. *Artificial Intelligence*, 23:13–27, 1984.

[6] N. Nilsson. *Principles of Artificial Intelligence*. Palo Alto, CA: Tioga Publishing, 1980.

[7] C. H. Papadimitriou and J. N. Tsitsiklis. The complexity of Markov decision processses. *Mathematics of Operations Research*, 12(3):441–450, 1987.